# Kernel Descriptors for Visual Recognition

**Liefeng Bo**
University of Washington
Seattle WA 98195, USA

**Xiaofeng Ren**
Intel Labs Seattle
Seattle WA 98105, USA

**Dieter Fox**
University of Washington & Intel Labs Seattle
Seattle WA 98195 & 98105, USA

## Abstract

The design of low-level image features is critical for computer vision algorithms. Orientation histograms, such as those in SIFT [16] and HOG [3], are the most successful and popular features for visual object and scene recognition. We highlight the kernel view of orientation histograms, and show that they are equivalent to a certain type of match kernels over image patches. This novel view allows us to design a family of kernel descriptors which provide a unified and principled framework to turn pixel attributes (gradient, color, local binary pattern, *etc.*) into compact patch-level features. In particular, we introduce three types of match kernels to measure similarities between image patches, and construct compact low-dimensional kernel descriptors from these match kernels using kernel principal component analysis (KPCA) [23]. Kernel descriptors are easy to design and can turn any type of pixel attribute into patch-level features. They outperform carefully tuned and sophisticated features including SIFT and deep belief networks. We report superior performance on standard image classification benchmarks: Scene-15, Caltech-101, CIFAR10 and CIFAR10-ImageNet.

## 1 Introduction

Image representation (features) is arguably the most fundamental task in computer vision. The problem is highly challenging because images exhibit high variations, are highly structured, and lie in high dimensional spaces. In the past ten years, a large number of low-level features over images have been proposed. In particular, orientation histograms such as SIFT [16] and HOG [3] are the most popular low-level features, essential to many computer vision tasks such as object recognition and 3D reconstruction. The success of SIFT and HOG naturally raises questions on how they measure the similarity between image patches, how we should understand the design choices in them, and whether we can find a principled way to design and learn comparable or superior low-level image features.

In this work, we highlight the kernel view of orientation histograms and provide a unified way to low-level image feature design and learning. Our low-level image feature extractors, kernel descriptors, consist of three steps: (1) design match kernels using pixel attributes; (2) learn compact basis vectors using kernel principle component analysis; (3) construct kernel descriptors by projecting the infinite-dimensional feature vectors to the learned basis vectors. We show how our framework is applied to gradient, color, and shape pixel attributes, leading to three effective kernel descriptors. We validate our approach on four standard image category recognition benchmarks, and show that our kernel descriptors surpass both manually designed and well tuned low-level features (SIFT) [16] and sophisticated feature learning approaches (convolutional networks, deep belief networks, sparse coding, *etc.*) [10, 26, 14, 24].

The most relevant work to this paper is that of efficient match kernels (EMK) [1], which provides a kernel view to the frequently used Bag-of-Words representation and forms image-level features by learning compact low dimensional projections or using random Fourier transformations. While the work on efficient match kernels is interesting, the hand-crafted SIFT features are still used as the basic building block. Another related work is based on mathematics of the neural response, which shows that the hierarchical architectures motivated by the neuroscience of the visual cortex is associated to the derived kernel [24]. Instead, the goal of this paper is to provide a deep understanding of how orientation histograms (SIFT and HOG) work, and we can generalize them and design novel low-level image features based on the kernel insight. Our kernel descriptors are general and provide a principled way to convert pixel attributes to patch-level features. To the best of our knowledge, this is the first time that low-level image features are designed and learned from scratch using kernel methods; they can serve as the foundation of many computer vision tasks including object recognition.

This paper is organized as follows. Section 2 introduces the kernel view of histograms. Our novel kernel descriptors are presented in Section 3, followed by an extensive experimental evaluation in Section 4. We conclude in Section 5.

## 2 Kernel View of Orientation Histograms

Orientation histograms, such as SIFT [16] and HOG [3], are the most commonly used low-level features for object detection and recognition. Here we describe the kernel view of such orientation histograms features, and show how this kernel view can help overcome issues such as orientation binning. Let $\theta(z)$ and $m(z)$ be the orientation and magnitude of the image gradient at a pixel $z$. In HOG and SIFT, the gradient orientation of each pixel is discretized into a $d-$dimensional indicator vector $\delta(z) = [\delta_1(z), \cdots, \delta_d(z)]$ with

$$\delta_i(z) = \begin{cases} 1, & \lfloor \frac{d\theta(z)}{2\pi} \rfloor = i - 1 \\ 0, & \text{otherwise} \end{cases} \tag{1}$$

where $\lfloor x \rfloor$ takes the largest integer less than or equal to $x$ (we will describe soft binning further below). The feature vector of each pixel $z$ is a weighted indicator vector $F(z) = m(z)\delta(z)$. Aggregating feature vectors of pixels over an image patch $P$, we obtain the histogram of oriented gradients:

$$F_h(P) = \sum_{z \in P} \widetilde{m}(z)\delta(z) \tag{2}$$

where $\widetilde{m}(z) = m(z)/\sqrt{\sum_{z \in P} m(z)^2 + \epsilon_g}$ is the normalized gradient magnitude, with $\epsilon_g$ a small constant. $P$ is typically a $4 \times 4$ rectangle in SIFT and an $8 \times 8$ rectangle in HOG. Without loss of generality, we consider $L2$-based normalization here. In object detection [3, 5] and matching based object recognition [18], linear support vector machines or the $L2$ distance are commonly applied to sets of image patch features. This is equivalent to measuring the similarity of image patches using a linear kernel in the feature map $F_h(P)$ in kernel space:

$$K_h(P, Q) = F_h(P)^\top F_h(Q) = \sum_{z \in P} \sum_{z' \in Q} \widetilde{m}(z)\widetilde{m}(z')\delta(z)^\top \delta(z') \tag{3}$$

where $P$ and $Q$ are patches usually from two different images. In Eq. 3, both $k_{\widetilde{m}}(z, z') = \widetilde{m}(z)\widetilde{m}(z')$ and $k_\delta(z, z') = \delta(z)^\top \delta(z')$ are the inner product of two vectors and thus are positive definite kernels. Therefore, $K_h(P, Q)$ is a match kernel over sets (here the sets are image patches) as in [8, 1, 11, 17, 7]. Thus Eq. 3 provides a kernel view of HOG features over image patches. For simplicity, we only use one image patch here; it is straightforward to extend to sets of image patches.

The hard binning underlying Eq. 1 is only for ease of presentation. To get a kernel view of soft binning [13], we only need to replace the delta function in Eq. 1 by the following, soft $\delta(\cdot)$ function:

$$\delta_i(z) = \max(\cos(\theta(z) - a_i)^9, 0) \tag{4}$$

where $a(i)$ is the center of the $i-$th bin. In addition, one can easily include soft spatial binning by normalizing gradient magnitudes using the corresponding spatial weights. The $L2$ distance between $P$ and $Q$ can be expressed as $D(P, Q) = 2 - 2F(P)^\top F(Q)$ as we know $F(P)^\top F(P) = 1$, and the kernel view can be provided in the same manner.

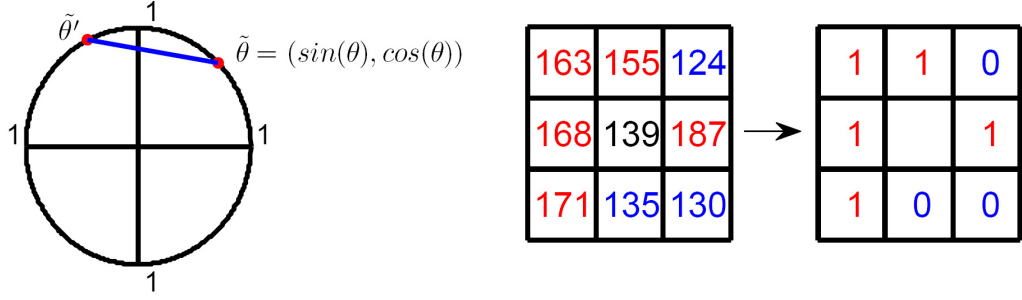

Figure 1: Pixel attributes. *Left*: Gradient orientation representation. To measure similarity between two pixel orientation gradients $\theta$ and $\theta'$, we use the $L2$ norm between the normalized gradient vectors $\widetilde{\theta} = [\sin(\theta)\cos(\theta)]$ and $\widetilde{\theta'} = [\sin(\theta')\cos(\theta')]$. The red dots represent the normalized gradient vectors, and the blue line represents the distance between them. *Right*: Local binary patterns. The values indicate brightness of pixels in a $3 \times 3$ patch. Red pixels have intensities larger than the center pixel, blue pixels are darker. The 8-dimensional indicator vector is the resulting local binary pattern.

Note that the kernel $k_{\widetilde{m}}(z, z')$ measuring the similarity of gradient magnitudes of two pixels is linear in gradient magnitude. $k_\delta(z, z')$ measures the similarity of gradient orientations of two pixels: 1 if two gradient orientations are in the same bin, and 0 otherwise (Eq.1, hard binning). As can be seen, this kernel introduces quantization errors and could lead to suboptimal performance in subsequent stages of processing. While soft binning results in a smoother kernel function, it still suffers from discretization. This motivates us to search for alternative match kernels which can measure the similarity of image patches more accurately.

## 3 Kernel Descriptors

### 3.1 Gradient, Color, and Shape Match Kernels

We introduce the following gradient match kernel, $K_{\text{grad}}$, to capture image variations:

$$K_{\text{grad}}(P, Q) = \sum_{z \in P} \sum_{z' \in Q} \widetilde{m}(z)\widetilde{m}(z')k_o(\widetilde{\theta}(z), \widetilde{\theta}(z'))k_p(z, z') \tag{5}$$

where $k_p(z, z') = \exp(-\gamma_p\|z - z'\|^2)$ is a Gaussian position kernel with $z$ denoting the 2D position of a pixel in an image patch (normalized to $[0, 1]$), and $k_o(\widetilde{\theta}(z), \widetilde{\theta}(z')) = \exp(-\gamma_o\|\widetilde{\theta}(z) - \widetilde{\theta}(z')\|^2)$ is a Gaussian kernel over orientations. To estimate the difference between orientations at pixels $z$ and $z'$, we use the following normalized gradient vectors in the kernel function $k_o$:

$$\widetilde{\theta}(z) = [\sin(\theta(z))\cos(\theta(z))] . \tag{6}$$

The $L2$ distance between such vectors measures the difference of gradient orientations very well (see Figure 1). Note that computing the $L2$ distance on the raw angle values $\theta$ instead of the normalized gradient vectors $\widetilde{\theta}$ would cause wrong similarity in some cases. For example, consider the two angles $2\pi - 0.01$ and $0.01$, which have very similar orientation but very large $L2$ distance.

To summarize, our gradient match kernel $K_{\text{grad}}$ consists of three kernels: the normalized linear kernel is the same as that in the orientation histograms, weighting the contribution of each pixel using gradient magnitudes; the orientation kernel $k_o$ computes the similarity of gradient orientations; and the position Gaussian kernel $k_p$ measures how close two pixels are spatially.

The kernel view of orientation histograms provides a simple, unified way to turn pixel attributes into patch-level features. One immediate extension is to construct color match kernels over pixel values:

$$K_{\text{col}}(P, Q) = \sum_{z \in P} \sum_{z' \in Q} k_c(c(z), c(z'))k_p(z, z') \tag{7}$$

where $c(z)$ is the pixel color at position $z$ (intensity for gray images and RGB values for color images). $k_c(c(z), c(z')) = \exp\left(-\gamma_c\|c(z) - c(z')\|^2\right)$ measures how similar two pixel values are.

While the gradient match kernel can capture image variations and the color match kernel can describe image appearance, we find that a match kernel over local binary patterns can capture local shape more effectively: [19]:

$$K_{\text{shape}}(P, Q) = \sum_{z \in P} \sum_{z' \in Q} \widetilde{s}(z)\widetilde{s}(z')k_b(b(z), b(z'))k_p(z, z') \tag{8}$$

where $\widetilde{s}(z) = s(z)/\sqrt{\sum_{z \in P} s(z)^2 + \epsilon_s}$, $s(z)$ is the standard deviation of pixel values in the $3 \times 3$ neighborhood around $z$, $\epsilon_s$ a small constant, and $b(z)$ is binary column vector binarizes the pixel value differences in a local window around $z$ (see Fig. 1(*right*)). The normalized linear kernel $\widetilde{s}(z)\widetilde{s}(z')$ weighs the contribution of each local binary pattern, and the Gaussian kernel $k_b(b(z), b(z')) = \exp(-\gamma_b\|b(z) - b(z')\|^2)$ measures shape similarity through local binary patterns.

Match kernels defined over various pixel attributes provide a unified way to generate a rich, diverse visual feature set, which has been shown to be very successful to boost recognition accuracy [6]. As validated by our own experiments, gradient, color and shape match kernels are strong in their own right and complement one another. Their combination turn out to be always (much) better than the best individual feature.

## 3.2   Learning Compact Features

Match kernels provide a principled way to measure the similarity of image patches, but evaluating kernels can be computationally expensive when image patches are large [1]. Both for computational efficiency and for representational convenience, we present an approach to extract the compact low-dimensional features from match kernels: (1) uniformly and densely sample sufficient basis vectors from support region to guarantee accurate approximation to match kernels; (2) learn compact basis vectors using kernel principal component analysis. An important advantage of our approach is that no local minima are involved, unlike constrained kernel singular value decomposition [1].

We now describe how our compact low-dimensional features are extracted from the gradient kernel $K_{\text{grad}}$; features for the other kernels can be generated the same way. Rewriting the kernels in Eq. 5 as inner products $k_o(\widetilde{\theta}(z), \widetilde{\theta}(z')) = \phi_o(\widetilde{\theta}(z))^\top \phi_o(\widetilde{\theta}(z'))$, $k_p(z, z') = \phi_p(z)^\top \phi_p(z')$, we can derive the following feature over image patches:

$$F_{\text{grad}}(P) = \sum_{z \in P} \widetilde{m}(z)\phi_o(\widetilde{\theta}(z)) \otimes \phi_p(z) \tag{9}$$

where $\otimes$ is the tensor product. For this feature, it follows that $F_{\text{grad}}(P)^\top F_{\text{grad}}(Q) = K_{\text{grad}}(P, Q)$. Because we use Gaussian kernels, $F_{\text{grad}}(P)$ is an infinite-dimensional vector.

A straightforward way to dimension reduction is to sample sufficient image patches from training images and perform KPCA for match kernels. However, such approach makes the learned features depend on the task at hand. Moreover, KPCA can become computationally infeasible when the number of patches is very large.

**Sufficient Finite-dimensional Approximation.** We present an approach to approximate match kernels directly without requiring any image. Following classic methods, we learn finite-dimensional features by projecting $F_{\text{grad}}(P)$ into a set of basis vectors. A key issue in this projection process is how to choose a set of basis vectors which makes the finite-dimensional kernel approximate well the original kernel. Since pixel attributes are low-dimensional vectors, we can achieve a very good approximation by sampling sufficient basis vectors using a fine grid over the support region. For example, consider the Gaussian kernel $k_o(\widetilde{\theta}(z), \widetilde{\theta}(z'))$ over gradient orientation. Given a set of basis vectors $\{\phi_o(x_i)\}_{i=1}^{d_o}$ where $x_i$ are sampled normalized gradient vectors, we can approximate a infinite-dimensional vector $\phi_o(\widetilde{\theta}(z))$ by its projection into the space spanned by the set of these $d_o$ basis vectors. Following the formulation in [1], such a procedure is equivalent to using a finite-dimensional kernel:

$$\widetilde{k}_o(\widetilde{\theta}(z), \widetilde{\theta}(z')) = k_o(\widetilde{\theta}(z), X)^\top \left[\mathbf{K_o}^{-1}\right]_{ij} k_o(\widetilde{\theta}(z'), X) = \left[\mathbf{G}k_o(\widetilde{\theta}(z), X)\right]^\top \left[\mathbf{G}k_o(\widetilde{\theta}(z'), X)\right] \tag{10}$$

where $k_o(\widetilde{\theta}(z), X) = [k_o(\widetilde{\theta}(z), x_1), \cdots, k_o(\widetilde{\theta}(z), x_{d_o})]^\top$ is a $d_o \times 1$ vector, $\mathbf{K_o}$ is a $d_o \times d_o$ matrix with $\mathbf{K_o}_{ij} = k_o(x_i, x_j)$, and $\mathbf{K_o}^{-1} = \mathbf{G}^\top \mathbf{G}$. The resulting feature map $\widetilde{\phi}_o(\widetilde{\theta}(z)) = \mathbf{G}k_o(\widetilde{\theta}(z), X)$

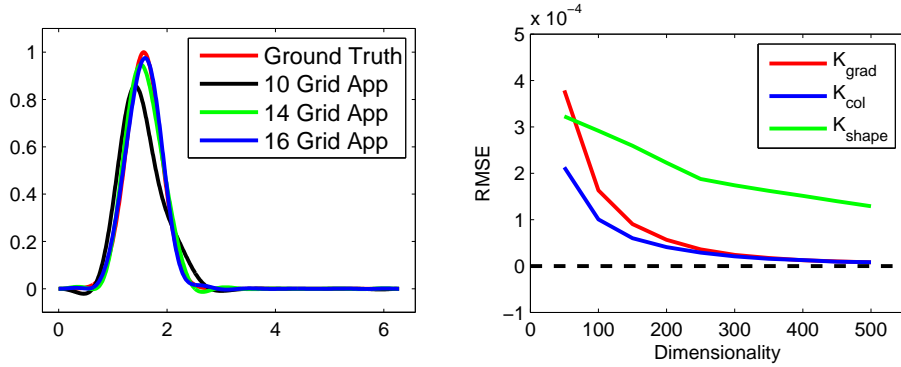

Figure 2: Finite dimensional approximation. *Left*: the orientation kernel $k_o(\widetilde{\theta}(z), \widetilde{\theta}(z'))$ and its finite-dimensional approximation. $\gamma_o$ is set to be 5 (as used in the experiments) and $\widetilde{\theta}(z')$ is fixed to [1 0]. All curves show kernel values as functions of $\widetilde{\theta}(z)$. The red line is the ground truth kernel function $k_o$, and the black, green and blue lines are the finite approximation kernels with different grid sizes. *Right*: root mean square error (RMSE) between KPCA approximation and the corresponding match kernel as a function of dimensionality. We compute the RMSE on randomly sampled 10000 datapoints. The three lines show the RMSE between the kernels $K_{\text{grad}}$ (red) and $K_{\text{col}}$ (blue) and $K_{\text{shape}}$ (green), and their respective approximation kernels.

is now only $d_o-$dimensional. In a similar manner, we can also approximate the kernels $k_p$, $k_c$ and $k_b$. The finite-dimensional feature for the gradient match kernel is $\widetilde{F}_{\text{grad}}(P) = \sum_{z \in P} \widetilde{m}(z) \widetilde{\phi}_o(\widetilde{\theta}(z)) \otimes \widetilde{\phi}_p(z)$, and may be efficiently used as features over image patches. We validate our intuition in Fig. 2. As we expect, the approximation error rapidly drops with increasing grid sizes. When the grid size is larger than 16, the finite kernel and the original kernel become virtually indistinguishable. For the shape kernel over local binary patterns, because the variables are binary, we simply choose the set of all $2^8 = 256$ basis vectors and thus no approximation error is introduced.

**Compact Features.** Although $\widetilde{F}_{\text{grad}}(P)$ is finite-dimensional, the dimensionality can be high due to the tensor product. For example, consider the shape kernel descriptor: the size of basis vectors on kernel $k_b$ is 256; if we choose the basis vectors of the position kernel $k_p$ on a $5 \times 5$ regular grid, the dimensionality of the resulting shape kernel descriptor $F_{\text{shape}}$ would be $256 \times 25 = 6400$, too high for practical purposes. Dense uniform sampling leads to accurate approximation but does not guarantee orthogonality of the basis vectors, thus introducing redundance. The size of basis vectors can be further reduced by performing kernel principal component analysis over joint basis vectors: $\{\phi_o(x_1) \otimes \phi_p(y_1), \cdots, \phi_o(x_{d_o}) \otimes \phi_p(y_{d_p})\}$, where $\phi_p(y_s)$ are basis vectors for the position kernel and $d_p$ is the number of basis vectors. The $t-$th kernel principal component can be written as

$$\text{PC}^t = \sum_{i=1}^{d_o} \sum_{j=1}^{d_p} \alpha_{ij}^t \phi_o(x_i) \otimes \phi_p(y_j) \tag{11}$$

where $d_o$ and $d_p$ are the sizes of basis vectors for the orientation and position kernel, respectively, and $\alpha_{ij}^t$ is learned through kernel principal component analysis: $\mathbf{K}_c \alpha^t = \lambda^t \alpha^t$, where $\mathbf{K}_c$ is a centered kernel matrix with $[\mathbf{K}_c]_{ijst} = k_o(x_i, x_j)k_p(y_s, y_t) - 2\sum_{i',s'} k_o(x_i, x_j)k_p(y_{s'}, y_t) + \sum_{i',j',s',t'} k_o(x_{i'}, x_{j'})k_p(y_{s'}, y_{t'})$. As shown in fig. (2), match kernels can be approximated rather accurately using the reduced basis vectors by KPCA. Under the framework of kernel principal component analysis, our gradient kernel descriptor (Eq. 5) has the form

$$\overline{F}_{\text{grad}}^t(P) = \sum_{i=1}^{d_o} \sum_{j=1}^{d_p} \alpha_{ij}^t \left\{ \sum_{z \in P} \widetilde{m}(z)k_o(\widetilde{\theta}(z), x_i)k_p(z, y_j) \right\} \tag{12}$$

The computational bottleneck of extracting kernel descriptors are to evaluate the kernel function $k_o k_p$ between pixels. Fortunately, we can compute two kernel values separately at the cost $d_o + d_p$, rather than $d_o d_p$. Our most expensive kernel descriptor, the shape kernel, takes about 4 seconds in MATLAB to compute on a typical image ($300 \times 300$ resolution and $16 \times 16$ image patches over

$8 \times 8$ grids). It is about 1.5 seconds for the gradient kernel descriptor, compared to about 0.4 seconds for SIFT under the same setting. A more efficient GPU-based implementation will certainly reduce the computation time for kernel descriptors such that real time applications become feasible.

## 4 Experiments

We compare gradient (KDES-G), color (KDES-C), and shape (KDES-S) kernel descriptors to SIFT and several other state of the art object recognition algorithms on four publicly available datasets: Scene-15, Caltech101, CIFAR10, and CIFAR10-ImageNet (a subset of ImageNet). For gradient and shape kernel descriptors and SIFT, all images are transformed into grayscale ($[0, 1]$) and resized to be no larger than $300 \times 300$ pixels with preserved ratio. Image intensity or RGB values are normalized to [0 1]. We extracted all low level features with $16 \times 16$ image patches over dense regular grids with spacing of 8 pixels. We used publicly available dense SIFT code at http://www.cs.unc.edu/ lazeb-nik [13], which includes spatial binning, soft binning and truncation (nonlinear cutoff at 0.2), and has been demonstrated to obtain high accuracy for object recognition. For our gradient kernel descriptors we use the same gradient computation as used for SIFT descriptors. We also evaluate the performance of the combination of the three kernel descriptors (KDES-A) by simply concatenating the image-level features vectors.

Instead of spatial pyramid kernels, we compute image-level features using efficient match kernels (EMK), which has been shown to produce more accurate quantization. We consider $1 \times 1$, $2 \times 2$ and $4 \times 4$ pyramid sub-regions (see [1]), and perform constrained kernel singular value decomposition (CKSVD) to form image-level features, using 1,000 visual words (basis vectors in CKSVD) learned by K-means from about 100,000 image patch features. We evaluate classification performance with accuracy averaged over 10 random training/testing splits with the exception of the CIFAR10 dataset, where we report the accuracy on the test set. We have experimented both with linear SVMs and Laplacian kernel SVMs and found that Laplacian kernel SVMs over efficient match kernel features are always better than linear SVMs (see (§4.2)). We use Laplacian kernel SVMs in our experiments (except for the tiny image dataset CIFAR10).

### 4.1 Hyperparameter Selection

We select kernel parameters using a subset of ImageNet. We retrieve 8 everyday categories from the ImageNet collection: apple, banana, box, coffee mug, computer keyboard, laptop, soda can and water bottle. We choose basis vectors for $k_o$, $k_c$, and $k_p$ from 25, $5 \times 5 \times 5$ and $5 \times 5$ uniform grids, respectively, which give sufficient approximations to the original kernels (see also Fig. 2). We optimize the dimensionality of KPCA and match kernel parameters jointly using exhaustive grid search. Our experiments suggest that the optimal parameter settings are $r = 200$ (dimensionality of kernel descriptors), $\gamma_o = 5$, $\gamma_c = 4$, $\gamma_b = 2$, $\gamma_p = 3$, $\epsilon_g = 0.8$ and $\epsilon_s = 0.2$ (fig. 3). In the following experiments, we will keep these values fixed, even though the performance may improve if considering task-dependent hyperparameter selection.

### 4.2 Benchmark Comparisons

**Scene-15.** Scene-15 is a popular scene recognition benchmark from [13] which contains 15 scene categories with 200 to 400 images in each. SIFT features have been extensively used on Scene-15. Following the common experimental setting, we train our models on 1,500 randomly selected images (100 images per category) and test on the rest. We report the averaged accuracy of SIFT, KDES-S, KDES-C, KDES-G, and KDES-A over 10 random training/test splits in Table 1. As we see, both gradient and shape kernel descriptors outperform SIFT with a margin. Gradient kernel descriptors and shape kernel descriptors have similar performance. It is not surprising that the intensity kernel descriptor has a lower accuracy, as all the images are grayscale. The combination of the three kernel descriptors further boosts the performance by about 2 percent. Another interesting finding is that Laplacian kernel SVMs are significantly better than linear SVMs, $86.7\%$.

In our recognition system, the accuracy of SIFT is $82.2\%$ compared to $81.4\%$ in spatial pyramid match (SPM). We also tried to replace SIFT features with our gradient and shape kernel descriptors in SPM, and both obtained $83.5\%$ accuracy, 2 percent higher than SIFT features. To our best knowledge, our gradient kernel descriptor alone outperforms the best published result $84.2\%$ [27].

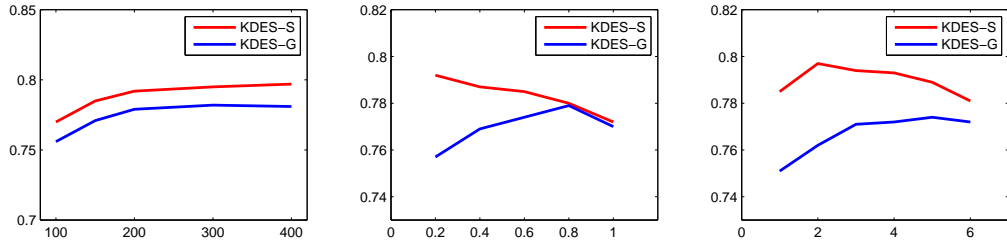

Figure 3: Hyperparameter selection. *left*: Accuracy as functions of feature dimensionality for orientation kernel (KDES-G) and shape kernel (KDES-S), respectively. *center*: Accuracy as functions of $\epsilon_g$ and $\epsilon_s$. *right*: Accuracy as function of $\gamma_o$ and $\gamma_b$.

| Methods | SIFT | KDES-C | KDES-G | KDES-S | KDES-A |
|---|---|---|---|---|---|
| Linear SVM | 76.7±0.7 | 38.5±0.4 | 81.6±0.6 | 79.8±0.5 | 81.9±0.6 |
| Laplacian kernel SVM | 82.2±0.9 | 47.9±0.8 | **85.0±0.6** | 84.9±0.7 | **86.7±0.4** |

Table 1: Comparisons of recognition accuracy on Scene-15: kernel descriptors and their combination vs SIFT.

**Caltech-101.** Caltech-101 [15] consists of 9,144 images in 101 object categories and one background category. The number of images per category varies from 31 to 800. Because many researchers have reported their results on Caltech-101, we can directly compare our algorithm to the existing ones. Following the standard experimental setting, we train classifiers on 30 images and test on no more than 50 images per category. We report our results in Table 2. We compare our kernel descriptors with recently published results obtained both by low-level feature learning algorithms, convolutional deep belief networks (CDBN), and sparse coding methods: invariant predictive sparse decomposition (IPSD) and locality-constrained linear coding. We observe that SIFT features in conjunction with efficient match kernels work well on this dataset and obtain 70.8% accuracy using a single patch size, which beat SPM with the same SIFT features by a large margin. Both our gradient kernel descriptor and shape kernel descriptor are superior to CDBN by a large margin.

We have performed feature extraction with three different patch sizes: $16 \times 16$, $25 \times 25$ and $31 \times 31$ and reached the same conclusions with many other researchers: multiple patch sizes (scales) can boost the performance by a few percent compared to the single patch size. Notice that both naive Bayesian nearest neighbor (NBNN) and locality-constrained linear coding should be compared to our kernel descriptors over multiple patch sizes because both of them used multiple scales to boost the performance. Using only our gradient kernel descriptor obtains 75.2% accuracy, higher than the results obtained by all other single feature based methods, to our best knowledge. Another finding is that the combination of three kernel descriptors outperforms any single kernel descriptor. We note that better performance has been reported with the use of more image features [6]. Our goal in this paper is to evaluate the strengths of kernel descriptors. To improve accuracy further, kernel descriptors can be combined with other types of image features.

**CIFAR-10.** CIFAR-10 is a labeled subset of the 80 million tiny images dataset [25, 12]. This dataset consists of 60,000 32x32 color images in 10 categories, with 5,000 images per category as training set and 1,000 images per category as test set. Deep belief networks have been extensively investigated on this dataset [21, 22]. We extract kernel descriptors over $8 \times 8$ image patches per pixel. Efficient match kernels over the three spatial grids $1 \times 1$, $2 \times 2$, and $3 \times 3$ are used to generate image-level features. The resulting feature vectors have a length of $(1+4+9)*1000$(visual words)$= 14000$ per kernel descriptor. Linear SVMs are trained due to the large number of training images.

| SPM [13] | 64.4±0.5 | kCNN [28] | 67.4 | KDES-C | 40.8±0.9 | KDES-C(M) | 42.4±0.5 |
|---|---|---|---|---|---|---|---|
| NBNN [2] | 73.0 | IPSD [10] | 56.0 | KDES-G | 73.3±0.6 | KDES-G(M) | **75.2±0.4** |
| CDBN [14] | 65.5 | LLC [26] | 73.4 ±0.5 | KDES-S | 68.2±0.7 | KDES-S(M) | 70.3±0.6 |
| SIFT | 70.8±0.8 | SIFT(M) | 73.2±0.5 | KDES-A | 74.5±0.8 | KDES-A(M) | **76.4±0.7** |

Table 2: Comparisons on Caltech-101. Kernel descriptors are compared to recently published results. (M) indicates that features are extracted with multiple image patch sizes.

| LR | 36.0 | GRBM, ZCAd images | 59.6 | mRBM | 59.7 | KDES-C | 53.9 |
|---|---|---|---|---|---|---|---|
| SVM | 39.5 | GRBM | 63.8 | cRBM | 64.7 | KDES-G | 66.3 |
| GIST[20] | 54.7 | fine-tuning GRBM | 64.8 | mcRBM | 68.3 | KDES-S | 68.2 |
| SIFT | 65.6 | GRBM two layers | 56.6 | mcRBM-DBN | 71.0 | KDES-A | **76.0** |

Table 3: Comparisons on CIFAR-10. Both logistic regression and SVMs are trained over image pixels.

| Methods | SIFT | KDES-C | KDES-G | KDES-S | KDES-A |
|---|---|---|---|---|---|
| Laplacian kernel SVMs | 66.5 ±0.4 | 56.4±0.8 | 69.0 ±0.8 | **70.5±0.7** | **75.2±0.7** |

Table 4: Comparisons on CIFAR10-ImageNet, subset of ImageNet using the 10 CIFAR categories.

We compare our kernel descriptors to deep networks [14, 9] and several baselines in table 3. One immediate observation is that sophisticated feature extractions are significantly better than raw pixel features. Linear logistic regression and linear SVMs over raw pixels only have accuracies of 36% and 39.5%, respectively, over 30 percent lower than deep belief networks and our kernel descriptors. SIFT features still work well on tiny images and have an accuracy of 65.2%. Color kernel descriptor, KDES-C, has 53.9% accuracy. This result is a bit surprising since each category has a large color variation. A possible explanation is that spatial information can help a lot. To validate our intuitions, we also evaluated the color kernel descriptor without spatial information (kernel features are extracted on $1 \times 1$ spatial grid), and only obtained 38.5% accuracy, 18 percent lower than the color kernel descriptor over pyramid spatial grids. KDES-G is slightly better than SIFT features. The shape kernel feature, KDES-S, has accuracy of 68.2%, and is the best single feature on this dataset. Combing the three kernel descriptors, we obtain the best performance of 76%, 5 percent higher than the most sophisticated deep network mcRBM-DBN, which model pixel mean and covariance jointly using factorized third-order Boltzmann machines.

**CIFAR-10-ImageNet.** Motivated by CIFAR-10, we collect a labeled subset of ImageNet [4] by retrieving 10 categories used in ImageNet: Airplane, Automobile, Bird, Cat, Deer, Dog, Frog, Horse, Ship and Truck. The total number of images is 15,561 with more than 1,200 images per category. This dataset is very challenging due to the following facts: multiple objects can appear in one image, only a small part of objects are visible, backgrounds are cluttered, and so on. We train models on 1,000 images per class and test on 200 images per category. We report the averaged results over 10 random training/test splits in Table 4. We can't finish running deep belief networks in a reasonable time since they are slow for running images of this scale. Both gradient and shape kernel descriptors achieve higher accuracy than SIFT features, which again confirms that our gradient kernel descriptor and shape kernel descriptor outperform SIFT features on high resolution images with the same category as CIFAR-10. We also ran the experiments on the downsized images, no larger than $50 \times 50$ with preserved ratio. We observe that the accuracy drops 4-6 percents compared to those on high resolution images. This validates that high resolution is helpful for object recognition.

## 5   Conclusion

We have proposed a general framework, *kernel descriptors*, to extract low-level features from image patches. Our approach is able to turn any pixel attribute into patch-level features in a unified and principled way. Kernel descriptors are based on the insight that the inner product of orientation histograms is a particular match kernel over image patches. We have performed extensive comparisons and confirmed that kernel descriptors outperform both SIFT features and hierarchical feature learning, where the former is the default choice for object recognition and the latter is the most popular low-level feature learning technique. To our best knowledge, we are the first to show how kernel methods can be applied for extracting low-level image features and show superior performance. This opens up many possibilities for learning low-level features with other kernel methods. Considering the huge success of kernel methods in the last twenty years, we believe that this direction is worth being pursued. In the future, we plan to investigate alternative kernels for low-level feature learning and learn pixel attributes from large image data collections such as ImageNet.

# References

[1] L. Bo and C. Sminchisescu. Efficient Match Kernel between Sets of Features for Visual Recognition. In *NIPS*, 2009.

[2] O. Boiman, E. Shechtman, and M. Irani. In defense of nearest-neighbor based image classification. In *CVPR*, 2008.

[3] N. Dalal and B. Triggs. Histograms of oriented gradients for human detection. In *CVPR*, 2005.

[4] J. Deng, W. Dong, R. Socher, L. Li, K. Li, and L. Fei-fei. ImageNet: A Large-Scale Hierarchical Image Database. In *CVPR*, 2009.

[5] P. Felzenszwalb, D. McAllester, and D. Ramanan. A discriminatively trained, multiscale, deformable part model. In *CVPR*, 2008.

[6] P. Gehler and S. Nowozin. On feature combination for multiclass object classification. In *ICCV*, 2009.

[7] K. Grauman and T. Darrell. The pyramid match kernel: discriminative classification with sets of image features. In *ICCV*, 2005.

[8] D. Haussler. Convolution kernels on discrete structures. Technical report, 1999.

[9] K. Jarrett, K. Kavukcuoglu, M. Ranzato, and Y. LeCun. What is the best multi-stage architecture for object recognition? In *ICCV*, 2009.

[10] K. Kavukcuoglu, M. Ranzato, R. Fergus, and Y. LeCun. Learning invariant features through topographic filter maps. In *CVPR*, 2009.

[11] R. Kondor and T. Jebara. A kernel between sets of vectors. In *ICML*, 2003.

[12] A. Krizhevsky. Learning multiple layers of features from tiny images. Technical report, 2009.

[13] S. Lazebnik, C. Schmid, and J. Ponce. Beyond bags of features: Spatial pyramid matching for recognizing natural scene categories. In *CVPR*, 2006.

[14] H. Lee, R. Grosse, R. Ranganath, and A. Ng. Convolutional deep belief networks for scalable unsupervised learning of hierarchical representations. In *ICML*, 2009.

[15] F. Li, R. Fergus, and P. Perona. One-shot learning of object categories. *IEEE PAMI*, 2006.

[16] D. Lowe. Distinctive image features from scale-invariant keypoints. *IJCV*, 60:91–110, 2004.

[17] S. Lyu. Mercer kernels for object recognition with local features. In *CVPR*, 2005.

[18] K. Mikolajczyk and C. Schmid. A performance evaluation of local descriptors. *IEEE PAMI*, 27(10):1615–1630, 2005.

[19] T. Ojala, M. Pietikäinen, and T. Mäenpää. Multiresolution gray-scale and rotation invariant texture classification with local binary patterns. *IEEE PAMI*, 24(7):971–987, 2002.

[20] A. Oliva and A. Torralba. Modeling the shape of the scene: A holistic representation of the spatial envelope. *IJCV*, 42(3):145–175, 2001.

[21] M. Ranzato, Krizhevsky A., and G. Hinton. Factored 3-way restricted boltzmann machines for modeling natural images. In *AISTATS*, 2010.

[22] M. Ranzato and G. Hinton. Modeling pixel means and covariances using factorized third-order boltzmann machines. In *CVPR*, 2010.

[23] B. Schölkopf, A. Smola, and K. Müller. Nonlinear component analysis as a kernel eigenvalue problem. *Neural Computation*, 10:1299–1319, 1998.

[24] S. Smale, L. Rosasco, J. Bouvrie, A. Caponnetto, and T. Poggio. Mathematics of the neural response. *Foundations of Computational Mathematics*, 10(1):67–91, 2010.

[25] A. Torralba, R. Fergus, and W. Freeman. 80 million tiny images: A large data set for nonparametric object and scene recognition. *IEEE PAMI*, 30(11):1958–1970, 2008.

[26] J. Wang, J. Yang, K. Yu, F. Lv, T. Huang, and Y. Guo. Locality-constrained linear coding for image classification. In *CVPR*, 2010.

[27] J. Wu and J. Rehg. Beyond the euclidean distance: Creating effective visual codebooks using the histogram intersection kernel. 2002.

[28] K. Yu, W. Xu, and Y. Gong. Deep learning with kernel regularization for visual recognition. In *NIPS*, 2008.

